# Operators and curried functions:
# Training and analysis of simple recurrent networks

**Janet Wiles**
Depts of Psychology and Computer Science,
University of Queensland
QLD 4072 Australia.
janetw@cs.uq.oz.au

**Anthony Bloesch,**
Dept of Computer Science,
University of Queensland,
QLD 4072 Australia
anthonyb@cs.uq.oz.au

## Abstract

We present a framework for programming the hidden unit representations of simple recurrent networks based on the use of *hint* units (additional targets at the output layer). We present two ways of analysing a network trained within this framework: Input patterns act as *operators* on the information encoded by the context units; symmetrically, patterns of activation over the context units act as *curried functions* of the input sequences. Simulations demonstrate that a network can learn to represent three different functions simultaneously and canonical discriminant analysis is used to investigate how operators and curried functions are represented in the space of hidden unit activations.

## 1 INTRODUCTION

Many recent papers have contributed to the understanding of recurrent networks and their potential for modelling sequential phenomena (see for example Giles, Sun, Chen, Lee, & Chen, 1990; Elman, 1989; 1990; Jordan, 1986; Cleeremans, Servan-Schreiber & McClelland, 1989; Williams & Zipser, 1988). Of particular interest in these papers is the development of recurrent architectures and learning algorithms able to solve complex problems. The perspective of the work we present here has many similarities with these studies, however, we focus on *programming* a recurrent network for a specific task, and hence provide appropriate sequences of inputs to learn the temporal component.

The function computed by a neural network is conventionally represented by its weights. During training, the task of a network is to learn a set of weights that causes the appropriate action (or set of context–specific actions) for each input pattern. However, in a network with recurrent connections, patterns of activation are also part of the function computed by a network. After training (when the weights have been fixed) each input pattern has a specific effect on the pattern of activation across the hidden and output units which is modulated by the current state of those units. That is, each input pattern is a context sensitive *operator* on the state of the system.

To illustrate this idea, we present a task in which many sequences of the form, {F, arg1, ..., argn} are input to a network, which is required to output the value of each function, F(arg1, ..., argn). The task is interesting since it illustrates how more than one function can be computed by the same network and how the function selected can be specified by the inputs. Viewing all the inputs (both function patterns, F, and argument patterns, argi) as operators allows us to analyse the effect of each input on the state of the network (the pattern of activation in the hidden and context units). From this perspective, the weights in the network can be viewed as an interpreter which has been programmed to carry out the operations specified by each input pattern.

We use the term *programming* intentionally, to convey the idea that the actions of each input pattern play a specific role in the processing of a sequence. In the simulations described in this paper, we use the simple recurrent network (*SRN*) proposed by Elman (1990). The art of programming enters the simulations in the use of extra target units, called *hints*, that are provided at the output layer. At each step in learning a sequence, hints specify all the information that the network must preserve in the hidden unit representation (the state of the system) in order to calculate outputs later in the sequence (for a discussion of the use of hints in training a recurrent network see Rumelhart, Hinton & Williams, 1986).

## 2  SIMULATIONS

Three different boolean functions and their arguments were specified as sub–sequences of patterns over the inputs to an *SRN*. The network was required to apply the function specified by the first pattern in each sequence to each of the subsequent arguments in turn. The functions provided were boolean functions of the current input and previous output, AND, OR and XOR (i.e., *exclusive-or*) and the arguments were arbitrary length strings of 0's and 1's. The context units were not reset between sub-sequences. An *SRN* with 3 input, 5 hidden, 5 context, 1 output and 5 hint units was trained using backpropagation with a momentum term. The 5 hint units at the output layer provided information about the boolean functions during training (via the backpropagation of errors), but not during testing. The network was trained on three data sets each containing 700 (ten times the number of weights in the network) randomly generated patterns, forming function and arguments sequences of average length 0.5, 2 and 4 arguments respectively. The network was trained for one thousand iterations on each training set.

### 2.1  RESULTS AND GENERALISATION

After training, the network correctly computed every pattern in the three training sets (using a closest match criterion for scoring the output) and also in a test set of sequences generated using the same statistics. Generalisation test data consisting of all possible sequences composed of each function and eight arguments, and long sequences each of 50 arguments also produced the correct output for every pattern in every sequence. To test

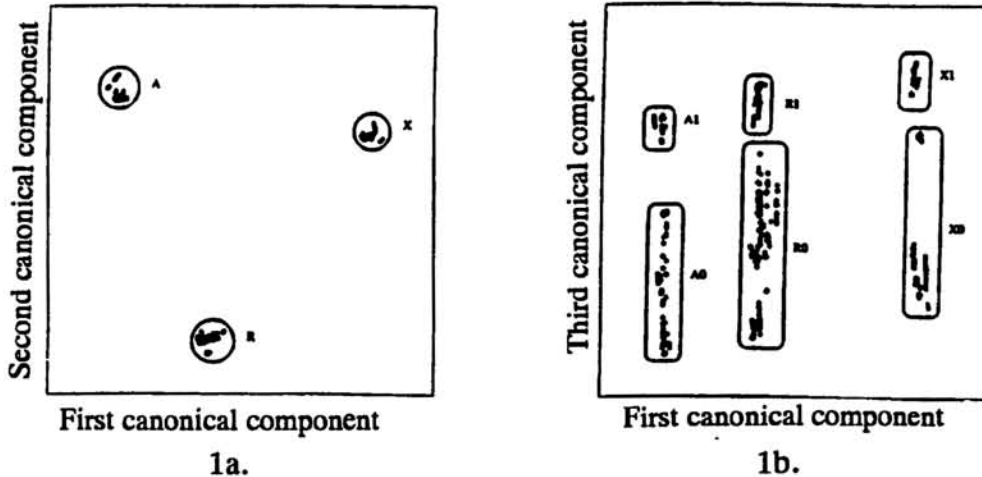

1a.                                1b.

Figure 1a.  The hidden unit patterns for the training data, projected onto the first two canonical components.  These components separate the patterns into 3 distinct regions corresponding to the initial pattern (AND, OR or XOR) in each sequence.  1b.  The first and third canonical components further separate the hidden unit patterns into 6 regions which have been marked in the diagrams above by the corresponding output classes A1, A0, R1, R0, X1 and X0.  These regions are effectively the computational states of the network.

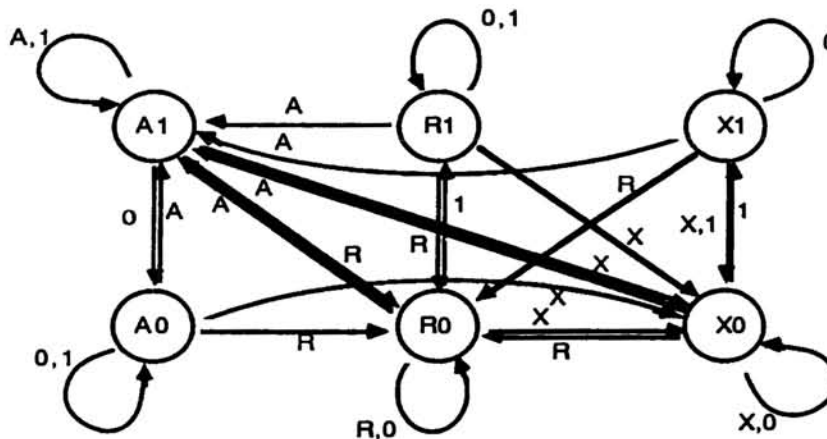

Figure 2.  Finite state machine to compute the three-function task.

Another way of considering sub–sequences in the input stream is to describe all the inputs as functions, not over the other inputs, as above, but as functions of the state (for which we use the term *operators*). Using this terminology, a sub–sequence is a composition of operators which act on the current state,

$$G(S(t)) = argt \circ ... \circ arg2 \circ arg1 \circ S(0),$$

where $(f \circ g)(x) = f(g(x))$, and $S(0)$ is the initial state of the network.  A consequence of describing the input patterns as operators is that even the 0 and 1 data bits can be seen as operators that transform the internal state (see Box 1).

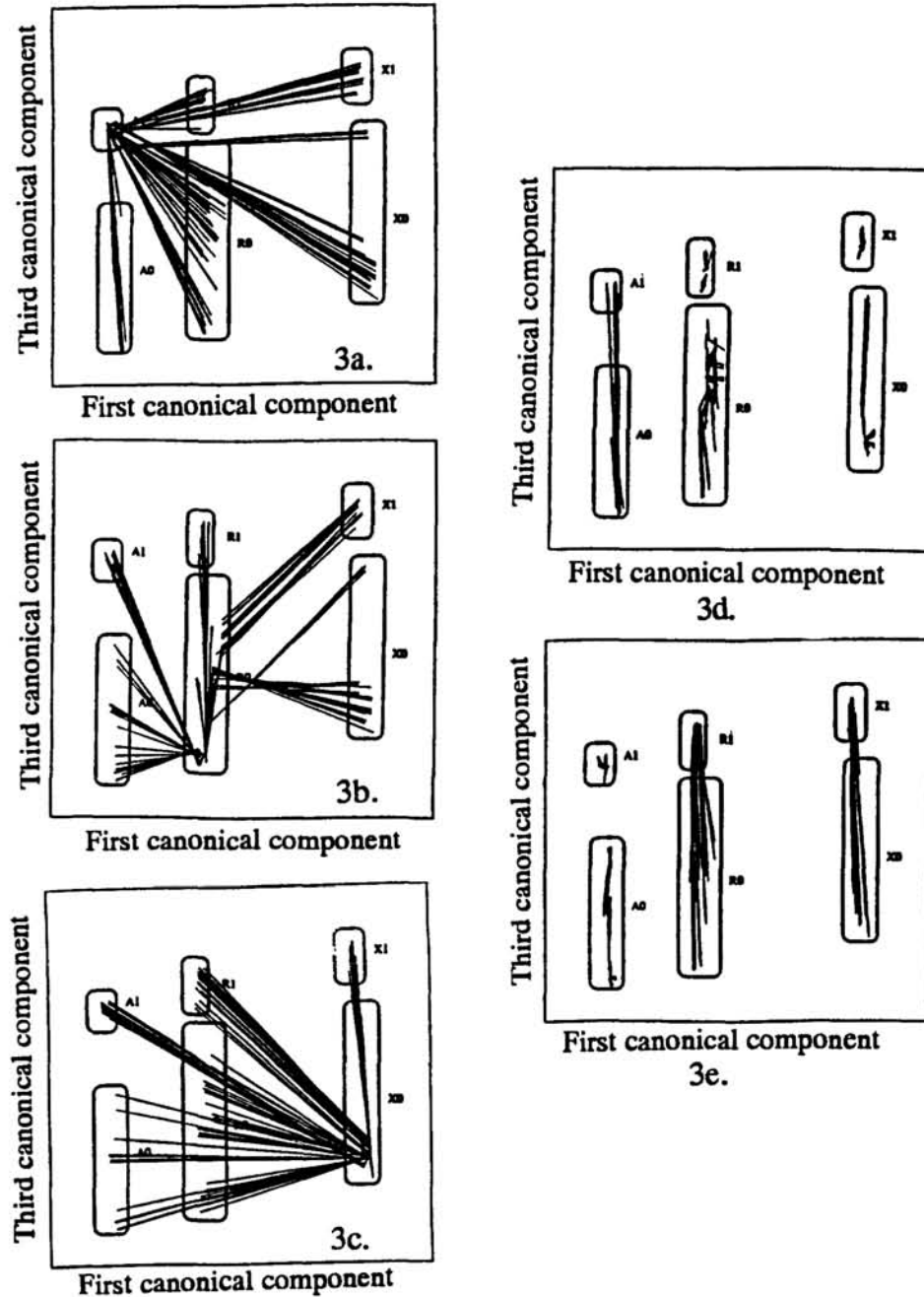

Figure 3. State transitions caused by each input pattern, projected onto the first and third canonical components of the hidden unit patterns (generated by the training data as in Figure 1). 3a-c. Transitions caused by the AND, OR and XOR input patterns respectively. From every point in the hidden unit space, the input patterns for AND, OR and XOR transform the hidden units to values corresponding to a point in the regions marked A1, R0 and X0 respectively. 3d-e. Transitions caused by the 0 and 1 input patterns respectively. The 0 and 1 inputs are context sensitive operators. The 0 input causes changes in the hidden unit patterns corresponding to transitions from the state A1 to A0, but does not cause transitions from the other 5 regions. Conversely, a 1 input does not cause the hidden unit patterns to change from the regions A1, A0 or R1, but causes transitions from the regions R0, X1 and X0.

| Input operators | Patterns on the input units | Effect on information encoded in the state | | |
|---|---|---|---|---|
| AND | 0 1 1 | $cf \rightarrow$ AND | | |
| OR | 1 1 0 | $cf \rightarrow$ OR | | |
| XOR | 1 0 1 | $cf \rightarrow$ XOR | | |
| 1 | 1 1 1 | $x(t) \rightarrow$ | $x(t\text{-}1)$ | if $cf$ = AND |
| | | | 1 | if $cf$ = OR |
| | | | NOT($x(t\text{-}1)$) | if $cf$ = XOR |
| 0 | 0 0 0 | $x(t) \rightarrow$ | 0 | if $cf$ = AND |
| | | | $x(t\text{-}1)$ | if $cf$ = OR |
| | | | $x(t\text{-}1)$ | if $cf$ = XOR |

Box 1. Operators for the 5 input patterns. The operation performed by each input pattern is described in terms of the effect it has on information encoded by the hidden unit patterns. The first and second columns specify the input operators and their corresponding input patterns. The third column specifies the effect that each input in a sub–sequence has on information encoded in the state, represented as $cf$, for current function, and $x(t)$ for the last output.

For each input pattern, we plotted all the transitions in hidden unit space resulting from that input projected onto the canonical components used in Figure 1. Figures 3a to 3e show transitions for each of the five input operators. For the three "function" inputs, OR, AND, and XOR, the effect is to collapse the hidden unit patterns to a single region – a particular state. These are relatively context insensitive operations. For the two "argument" inputs, 0 and 1, the effect is sensitive to the context in which the input occurs (i.e., the previous state of the hidden units). A similar analysis of the states themselves focuses on the hidden unit patterns and the information that they must encode in order to compute the three-function task. At each timestep the weights in the network construct a pattern of activation over the hidden units that reduces the structured arguments of a complex function of several arguments by a simpler function of one less argument. This can be represented as follows:

$$G(F, arg1, ... argn) \rightarrow F(arg1, ... argn)$$
$$\rightarrow F_{arg1}(arg2, ... argn)$$
$$\rightarrow F_{arg1arg2}(arg3, ... argn).$$

This process of replacing structured arguments by a corresponding sequence of simple ones is known as *currying* the input sequence (for a review of curried functions, see Bird and Wadler, 1988). Using this terminology, the pattern of activation in the hidden units is a *curried function* of the entire input sequence up to that time step. The network combines the previous hidden unit patterns (preserved in the context units) with the current input patterns to compute the next curried function in the sequence. Since there are 6 states required by the network, there are 6 classes of equivalent curried functions. Figure 4 shows the transition diagrams for each of the 6 equivalence classes of curried functions from the same simulation shown in Figures 1 and 3.

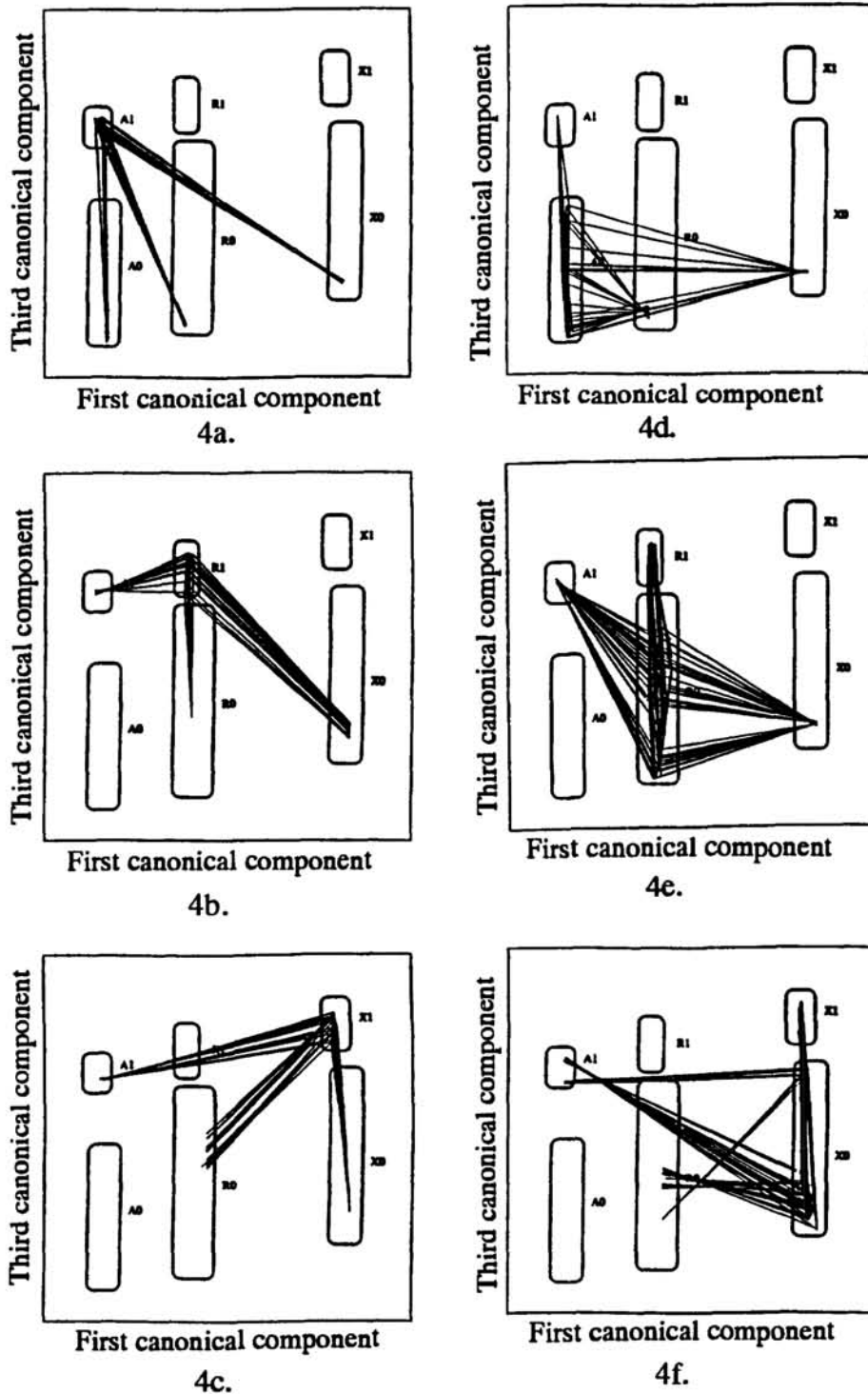

Figure 4. State transitions for each hidden unit pattern, grouped into classes of curried functions, projected onto the first and third canonical components. 4a-f. Transitions from A1, R1, X1, A0, R0 and X0 respectively. Each pattern of activation corresponds to a curried function of the input sequence up to that item in the sequence.

how often the network finds a good solution, five simulations were completed with the above parameters, all started with different sets of random weights, and randomly generated training patterns. Three simulations learnt the training set perfectly (the other two simulations appeared to be converging, but slowly: worst case error less than 1%). On the test data, the results were also good (worst case 7% error).

## 2.2 ANALYSIS

The hidden unit patterns generated by the training data in the simulations described above were analysed using canonical discriminant analysis (*CDA*, Kotz & Johnson, 1982). Six output classes were specified, corresponding to one class for each output for each function. The output classes were used to compute the first three canonical components of the hidden unit patterns (which are 5–dimensional patterns corresponding to the 5 hidden units). The graph of the first two canonical components (see Figure 1a) shows the hidden unit patterns separated into three tight clusters, corresponding to the sequence type (OR, AND and XOR). The first and third canonical components (see Figure 1b) reveals more of the structure within each class. The six classes of hidden unit patterns are spread across six distinct regions (these correspond to the 6 states of the minimal finite state machine, as shown in Figure 2). The first canonical component separates the hidden unit patterns into sequence type (OR, AND, or XOR, separated across the page). Within each region, the third canonical component separates the outputs into 0's and 1's (separated down the page). Cluster analysis followed by *CDA* on the clusters gave similar results.

# 3  DISCUSSION

In a network that is dedicated to computing a boolean function such as XOR, it seems obvious that the information for computing the function is in the weights. The simulations described in this paper show that this intuition does not necessarily generalise to other networks. The three-function task requires that the network use the first input in a sequence to select a function which is then applied to subsequent arguments. In general, for any given network, the function that is computed over a given sub–sequence will be specified by the interaction between the weights and the activation pattern.

The function computed by the networks in these simulations can be described in terms of the output of the global function, $O(t) = G(arg1, ..., argt)$, computed by the weights of the network, which is a function of the whole input sequence. An equivalent description can be given in terms of sub–sequences of the input stream, which specify a boolean function over subsequent arguments, $G(F, arg1, ..., argt) = F(arg1, ..., argt)$. Both these levels of description follow the traditional approach of separating functions and data, where the patterns of activity can be described as *either* one or the other.

It appears to us that descriptions based on operators and curried functions provide a promising approach for the integration of representation and process within recurrent networks. For example, in the simulations described by Elman (1990), words can be understood as denoting operators which act on the state of the recurrent network, rather than denoting objects as they do in traditional linguistic theory. The idea of currying can also be applied to feedback from the output layer, for example in the networks developed by Jordan (1986), or to the product units used by Giles et al. (1990).

## Acknowledgements

We thank Jeff Elman, Ian Hayes, Julie Stewart and Bill Wilson for many discussions on these ideas, and Simon Dennis and Steven Phillips for developing the canonical discriminant program. This work was supported by grants from the Australian Research Council and A. Bloesch was supported by an Australian Postgraduate Research Award.

## References

Bird, R., and Wadler P. (1988). *Introduction to Functional Programming*, Prentice Hall, NY.

Cleeremans, A., Servan-Schreiber, D., and McClelland, J.L. (1989). Finite state automata and simple recurrent networks, *Neural Computation*, 1, 372-381.

Elman, J. (1989). Representation and structure in connectionist models. UCSD CRL Technical Report 8903, August 1989.

Elman, J. (1990). Finding structure in time. *Cognitive Science*, 14, 179-211.

Giles, C. L., Sun, G. Z., Chen, H. H., Lee, Y. C., and Chen, D. (1990). Higher Order Recurrent Networks. In D.S. Touretzky (ed.) Advances in Neural Information Processing Systems 2, Morgan-Kaufmann, San Mateo, Ca., 380-387.

Jordan, M. I. (1986). Serial order: A parallel distributed processing approach. Institute for Cognitive Science, Technical Report 8604. UCSD.

Kotz, S., and Johnson, N.L. (1982). *Encyclopedia of Statistical Sciences*. John Wiley and Sons, NY.

Rumelhart, D.E., Hinton, G.E., and Williams, R.J. (1986). Learning internal representations by error propagation. In D.E. Rumelhart & J.L. McClelland (eds.), *Parallel distributed processing: Explorations in the microstructure of cognition* (Vol. 1, pp. 318-362). Cambridge, MA: MIT Press.

Williams, R. J., and Zipser, D. (1988). A Learning Algorithm for Continually Running Fully Recurrent Neural Networks, Institute for Cognitive Science, Technical Report 8805. UCSD.